# Simultaneously Leveraging Output and Task Structures for Multiple-Output Regression

**Piyush Rai**[†]
Dept. of Computer Science
University of Texas at Austin
Austin, TX
piyush@cs.utexas.edu

**Abhishek Kumar**[†]
Dept. of Computer Science
University of Maryland
College Park, MD
abhishek@cs.umd.edu

**Hal Daumé III**
Dept. of Computer Science
University of Maryland
College Park, MD
hal@umiacs.umd.edu

## Abstract

Multiple-output regression models require estimating multiple parameters, one for each output. Structural regularization is usually employed to improve parameter estimation in such models. In this paper, we present a multiple-output regression model that leverages the covariance structure of the latent model parameters as well as the *conditional* covariance structure of the observed outputs. This is in contrast with existing methods that usually take into account only one of these structures. More importantly, unlike some of the other existing methods, none of these structures need be known *a priori* in our model, and are *learned* from the data. Several previously proposed structural regularization based multiple-output regression models turn out to be special cases of our model. Moreover, in addition to being a rich model for multiple-output regression, our model can also be used in estimating the graphical model structure of a set of variables (multivariate outputs) *conditioned* on another set of variables (inputs). Experimental results on both synthetic and real datasets demonstrate the effectiveness of our method.

## 1 Introduction

Multivariate response prediction, also known as multiple-output regression [3] when the responses are real-valued vectors, is an important problem in machine learning and statistics. The goal in multiple-output regression is to learn a model for predicting $K > 1$ real-valued responses (the output) from $D$ predictors or features (the input), given a training dataset consisting of $N$ input-output pairs. Multiple-output prediction is also an instance of the problem of multitask learning [5, 10] where predicting each output is a task and all the tasks share the same input data. Multiple-output regression problems are encountered frequently in various application domains. For example, in computational biology [11], we often want to predict the gene-expression levels of multiple genes based on a set of single nucleotide polymorphisms (SNPs); in econometrics [17], we often want to predict the stock prices in the future using relevant macro-economic variables and stock prices in the past as inputs; in geostatistics, we are often interested in jointly predicting the concentration levels of different heavy metal pollutants [9]; and so on.

One distinguishing aspect of multiple-output regression is that the outputs are often related to each other via some underlying (and often *a priori* unknown) structure. A part of this can be captured by the imposing a relatedness structure among the regression coefficients (e.g., the weight vectors in a linear regression model) of all the outputs. We refer to the relatedness structure among the regression coefficients as *task structure*. However, there can still be some structure left in the outputs that is not explained by the regression coefficients alone. This can be due to a limited expressive power of our chosen hypothesis class (e.g., linear predictors considered in this paper). The residual structure that is left out when conditioned on inputs will be referred to as *output structure* here. This can be also be seen as the covariance structure in the output noise. It is therefore desirable to *simultaneously* learn

---
[†]Contributed equally

and leverage *both* the output structure and the task structure in multiple-output regression models for improved parameter estimation and prediction accuracy.

Although some of the existing multiple-output regression models have attempted to incorporate such structures [17, 11, 13], most of these models are restrictive in the sense that (1) they usually exploit only one of the two structures (output structure *or* task structure, but not both), and (2) they assume availability of prior information about such structures which may not always be available. For example, Multivariate Regression with Covariance Estimation [17] (MRCE) is a recently proposed method which *learns* the output structure (in form of the covariance matrix for correlated noise across multiple outputs) along with the regression coefficients (i.e., the weight vector) for predicting each output. However MRCE does not explicitly model the relationships among the regression coefficients of the multiple tasks and therefore fails to account for the task structure. More recently, [14] proposed an extension of the MRCE model by allowing weighting the individual entries of the regression coefficients and the entries of the output (inverse) covariance matrix, but otherwise this model has essentially the same properties as MRCE. Among other works, Graph-guided Fused Lasso [11] (GFlasso) incorporates task structure to some degree by assuming that the regression coefficients of all the outputs have similar sparsity patterns. This amounts to assuming that all the outputs share almost same set of relevant features. However, GFlasso assumes that output graph structure is known which is rarely true in practice. Some other methods such as[13] take into account the task structure by imposing structural sparsity on the regression coefficients of the multiple tasks but again assume that output structure is known *a priori* and/or is of a specific form. In [22], the authors proposed a multitask learning model by explicitly modeling the task structures as the task covariance matrix but this model does not take into account the output structure which is important in multiple-output regression problems.

In this paper, we present a multiple-output regression model that allows leveraging both output structure and task structure without assuming an *a priori* knowledge of either. In our model, both output structure and task structure are *learned* from the data, along with the regression coefficients for each task. Specifically, we model the output structure using the (inverse) covariance matrix of the correlated noise across the multiple outputs, and the task structure using the (inverse) covariance matrix of the regression coefficients of the multiple tasks being learned in the model. By explicitly modeling and learning the output structure and task structure, our model also addresses the limitations of the existing models that typically assume certain specific type of output structures (e.g., tree [13]) or task structures (e.g., shared sparsity [11]). In particular, a model with task relatedness structure based on shared sparsity on the task weight vectors may not be appropriate in many real applications where all the features are important for prediction and the true task structure is at a more higher level (e.g., weight vectors for some tasks are closer to each other compared to others). Apart from providing a flexible way of learning multiple-output regression, our model can also be used for the problem of conditional inverse covariance estimation of the (multivariate) outputs that depend on another set of inputs variables, an important problem that has been gaining significant attention recently [23, 15, 20, 4, 7, 6].

## 2 Multiple-Output Regression

In multiple-output regression, each input is associated with a *vector* of responses and the goal is the learn the input-output relationship given some training data consisting of input-output pairs. Formally, given an $N \times D$ input matrix $\mathbf{X} = [\mathbf{x}_1, \ldots, \mathbf{x}_N]^\top$ and an $N \times K$ output matrix $\mathbf{Y} = [\mathbf{y}_1, \ldots, \mathbf{y}_N]^\top$, the goal in multiple-output regression is to learn the functional relationship between the inputs $\mathbf{x}_n \in \mathbb{R}^D$ and the outputs $\mathbf{y}_n \in \mathbb{R}^K$. For a linear regression model, we write:

$$\mathbf{y}_n = \mathbf{W}^\top \mathbf{x}_n + \mathbf{b} + \epsilon_n \qquad \forall n = 1, \ldots, N \qquad (1)$$

Here $\mathbf{W} = [\mathbf{w}_1, \ldots, \mathbf{w}_K]$ denotes the $D \times K$ matrix where $\mathbf{w}_k$ denotes the regression coefficient of the $k$-th output, $\mathbf{b} = [b_1, \ldots, b_K]^\top \in \mathbb{R}^K$ is a vector of bias terms for the $K$ outputs, and $\epsilon_n = [\epsilon_{n1}, \ldots, \epsilon_{nK}]^\top \in \mathbb{R}^K$ is a vector consisting of the noise for each of the $K$ outputs. The noise is typically assumed to be Gaussian with a zero mean and uncorrelated across the $K$ outputs.

Standard parameter estimation for Equation 1 involves maximizing the (penalized) log-likelihood of the model, or equivalently minimizing the (regularized) loss function over the training data:

$$\arg\min_{\mathbf{W}, \mathbf{b}} \operatorname{tr}((\mathbf{Y} - \mathbf{X}\mathbf{W} - \mathbf{1}\mathbf{b}^\top)(\mathbf{Y} - \mathbf{X}\mathbf{W} - \mathbf{1}\mathbf{b}^\top)^\top) + \lambda R(\mathbf{W}) \qquad (2)$$

where $\text{tr}(.)$ denotes matrix trace, $\mathbf{1}$ an $N \times 1$ vector of all 1s and $R(\mathbf{W})$ the regularizer on the weight matrix $\mathbf{W}$ consisting of the regression weight vectors of all the outputs. For a choice of $R(\mathbf{W}) = \text{tr}(\mathbf{W}^\top \mathbf{W})$ (the $\ell_2$-squared norm, equivalent to assuming independent, zero-mean Gaussian priors on the weight vectors), solving Equation 2 amounts to solving $K$ independent regression problems and this solution ignores any correlations among the outputs or among the weight vectors.

## 3  Multiple-Output Regression with Output and Task Structures

To take into account both *conditional* output covariance and the covariance among the weight vectors $\mathbf{W} = [\mathbf{w}_1, \ldots, \mathbf{w}_K]$, we assume a full covariance matrix $\mathbf{\Omega}$ of size $K \times K$ on the output noise distribution to capture *conditional* output covariance, and a structured prior distribution on the weight vector matrix $\mathbf{W}$ that induces structural regularization of $\mathbf{W}$. We place the following prior distribution on $\mathbf{W}$

$$p(\mathbf{W}) \propto \prod_{k=1}^{K} \mathcal{N}or(\mathbf{w}_k|0, \mathbf{I}_D) \mathcal{MN}_{D \times K}(\mathbf{W}|\mathbf{0}_{D \times K}, \mathbf{I}_D \otimes \mathbf{\Sigma}) \qquad (3)$$

where $\mathcal{MN}_{D \times K}(\mathbf{M}, \mathbf{A} \otimes \mathbf{B})$ denotes the matrix-variate normal distribution with $\mathbf{M} \in \mathbb{R}^{D \times K}$ being its mean, $\mathbf{A} \in \mathbb{R}^{D \times D}$ its row-covariance matrix and $\mathbf{B} \in \mathbb{R}^{K \times K}$ its column-covariance matrix. Here $\otimes$ denotes the Kronecker product. In this prior distribution, the $\mathcal{N}or(\mathbf{w}_k|0, \mathbf{I}_D)$ factors regularize the weight vectors $\mathbf{w}_k$ *individually*, and the $\mathcal{MN}_{D \times K}(\mathbf{W}|\mathbf{0}_{D \times K}, \mathbf{I}_D \otimes \mathbf{\Sigma})$ term couples the $K$ weight vectors, allowing them to share statistical strength.

To derive our objective function, we start by writing down the likelihood of the model, for a set of $N$ i.i.d. observations:

$$\prod_{n=1}^{N} p(\mathbf{y}_n|\mathbf{x}_n, \mathbf{W}, \mathbf{b}) = \prod_{n=1}^{N} \mathcal{N}or(\mathbf{y}_n|\mathbf{W}^\top \mathbf{x}_n + \mathbf{b}, \mathbf{\Omega}) \qquad (4)$$

In the above, a diagonal $\mathbf{\Omega}$ would imply that the $K$ outputs are all *conditionally* independent of each other. In this paper, we assume a full $\mathbf{\Omega}$ which will allow us to capture the conditional output correlations.

Combining the prior on $\mathbf{W}$ and the likelihood, we can write down the posterior distribution of $\mathbf{W}$:

$$p(\mathbf{W}|\mathbf{X}, \mathbf{Y}, \mathbf{b}, \mathbf{\Omega}, \mathbf{\Sigma}) \quad \propto \quad p(\mathbf{W}) \prod_{n=1}^{N} p(\mathbf{y}_n|\mathbf{x}_n, \mathbf{W}, \mathbf{b})$$
$$= \prod_{k=1}^{K} \mathcal{N}or(\mathbf{w}_k|0, \mathbf{I}_D) \, \mathcal{MN}_{D \times K}(\mathbf{W}|\mathbf{0}_{D \times K}, \mathbf{I}_D \otimes \mathbf{\Sigma}) \, \prod_{n=1}^{N} \mathcal{N}or(\mathbf{y}_n|\mathbf{W}^\top \mathbf{x}_n + \mathbf{b}, \mathbf{\Omega})$$

Taking the log of the above and simplifying the resulting expression, we can then write the negative log-posterior of $\mathbf{W}$ as (ignoring the constants):

$$\text{tr}((\mathbf{Y} - \mathbf{XW} - \mathbf{1}\mathbf{b}^\top)\mathbf{\Omega}^{-1}(\mathbf{Y} - \mathbf{XW} - \mathbf{1}\mathbf{b}^\top)^\top) + N \log |\mathbf{\Omega}| + \text{tr}(\mathbf{W}\mathbf{W}^\top)$$
$$+ \text{tr}(\mathbf{W}\mathbf{\Sigma}^{-1}\mathbf{W}^\top) + D \log |\mathbf{\Sigma}|$$

where $\mathbf{1}$ denotes a $N \times 1$ vector of all 1s. Note that in the term $\text{tr}(\mathbf{W}\mathbf{\Sigma}^{-1}\mathbf{W}^\top)$, the inverse covariance matrix $\mathbf{\Sigma}^{-1}$ plays the role of coupling pairs of weight vectors, and therefore controls the amount of sharing between any pair of tasks. The task covariance matrix $\mathbf{\Sigma}$ as well as the conditional output covariance matrix $\mathbf{\Omega}$ will be learned from the data. For reasons that will become apparent later, we parameterize our model in terms of the *inverse* covariance matrices $\mathbf{\Omega}^{-1}$ and $\mathbf{\Sigma}^{-1}$ instead of covariance matrices. With this parameterization, the negative log-posterior becomes:

$$\text{tr}((\mathbf{Y} - \mathbf{XW} - \mathbf{1}\mathbf{b}^\top)\mathbf{\Omega}^{-1}(\mathbf{Y} - \mathbf{XW} - \mathbf{1}\mathbf{b}^\top)^\top) - N \log |\mathbf{\Omega}^{-1}| + \text{tr}(\mathbf{W}\mathbf{W}^\top)$$
$$+ \text{tr}(\mathbf{W}\mathbf{\Sigma}^{-1}\mathbf{W}^\top) - D \log |\mathbf{\Sigma}^{-1}| \qquad (5)$$

The objective function in Equation 5 naturally imposes positive-definite constraints on the inverse covariance matrices $\mathbf{\Omega}^{-1}$ and $\mathbf{\Sigma}^{-1}$. In addition, we will impose sparsity constraints (via an $\ell_1$ penalty) on $\mathbf{\Omega}^{-1}$ and $\mathbf{\Sigma}^{-1}$. Sparsity on these parameters is appealing in this context for two reasons: (1) Sparsity leads to improved robust estimates [19, 8] of $\mathbf{\Omega}^{-1}$ and $\mathbf{\Sigma}^{-1}$, and (2) Sparsity supports the notion that the output correlations and the task correlations tend to be sparse [21, 4, 8]

– not all pairs of outputs are related (given the inputs and other outputs), and likewise not all task pairs (and therefore the corresponding weight vectors) are related. Finally, we will also introduce regularization hyperparameters to control the trade-off between data-fit and model complexity. Parameter estimation in the model involves *minimizing* the negative log-posterior which is equivalent to minimizing the (regularized) loss function. The minimization problem is given as

$$\underset{\mathbf{W},\mathbf{b},\mathbf{\Sigma}^{-1},\mathbf{\Omega}^{-1}}{\arg\min} \quad \mathrm{tr}((\mathbf{Y} - \mathbf{XW} - \mathbf{1b}^{\top})\mathbf{\Omega}^{-1}(\mathbf{Y} - \mathbf{XW} - \mathbf{1b}^{\top})^{\top}) - N\log|\mathbf{\Omega}^{-1}| + \lambda\,\mathrm{tr}(\mathbf{WW}^{\top})$$

$$+ \lambda_1\,\mathrm{tr}(\mathbf{W\Sigma}^{-1}\mathbf{W}^{\top}) - D\log|\mathbf{\Sigma}^{-1}| + \lambda_2||\mathbf{\Omega}^{-1}||_1 + \lambda_3||\mathbf{\Sigma}^{-1}||_1 \tag{6}$$

where $||\mathbf{A}||_1$ denotes the sum of absolute values of the matrix $\mathbf{A}$. Note that by replacing the regularizer $\mathrm{tr}(\mathbf{WW}^{\top})$ with a sparsity inducing regularizer on the individual weight vectors $\mathbf{w}_1, \ldots, \mathbf{w}_K$, one can also learn Lasso-like sparsity [19] in the regression weights. In this exposition, however, we consider $\ell_2$ regularization on the regression weights and let the $\mathrm{tr}(\mathbf{W\Sigma}^{-1}\mathbf{W}^{\top})$ term capture the similarity between the weights of two tasks by learning the task inverse covariance matrix $\mathbf{\Sigma}^{-1}$. The above cost function is not jointly convex in the variables but is individually convex in each variable when others are fixed. We adopt an alternating optimization strategy that was empirically observed to converge in all our experiments. More details are provided in the experiments section. Finally, although it is not the main goal of this paper, since our model provides an estimate of the inverse covariance structure $\mathbf{\Omega}^{-1}$ of the outputs conditioned on the inputs, it can also be used for the more general problem of estimating the conditional inverse covariance [23, 15, 20, 4, 7] of a set of variables $\mathbf{y} = \{y_1, \ldots, y_K\}$ conditioned on another set of variables $\mathbf{x} = \{x_1, \ldots, x_D\}$, given *paired* samples of the form $\{(\mathbf{x}_1, \mathbf{y}_1), \ldots, (\mathbf{x}_N, \mathbf{y}_N)\}$.

### 3.1 Special Cases

In this section, we show that our model subsumes/generalizes some previously proposed models for multiple-output regression. Some of these include:

- **Multivariate Regression with Covariance Estimation (MRCE-$\ell_2$):** With the task inverse covariance matrix $\mathbf{\Sigma}^{-1} = \mathbf{I}_K$ and the bias term set to zero, our model results in the $\ell_2$ regularized weights variant of the MRCE model [17] which would be equivalent to minimizing the following objective:

$$\underset{\mathbf{W},\mathbf{\Omega}^{-1}}{\arg\min}\ \mathrm{tr}((\mathbf{Y} - \mathbf{XW})\mathbf{\Omega}^{-1}(\mathbf{Y} - \mathbf{XW})^{\top}) + \lambda\,\mathrm{tr}(\mathbf{WW}^{\top}) - N\log|\mathbf{\Omega}^{-1}| + \lambda_2||\mathbf{\Omega}^{-1}||_1$$

- **Multitask Relationship Learning for Regression (MTRL):** With the output inverse covariance matrix $\mathbf{\Omega}^{-1} = \mathbf{I}_K$ and the sparsity constraint on $\mathbf{\Sigma}^{-1}$ dropped, our model results in the regression version of the multitask relationship learning model proposed in [22]. Specifically, the corresponding objective function would be:

$$\underset{\mathbf{W},\mathbf{\Sigma}^{-1}}{\arg\min}\ \mathrm{tr}((\mathbf{Y} - \mathbf{XW})(\mathbf{Y} - \mathbf{XW})^{\top}) + \lambda\,\mathrm{tr}(\mathbf{WW}^{\top}) + \lambda_1\,\mathrm{tr}(\mathbf{W\Sigma}^{-1}\mathbf{W}^{\top}) - D\log|\mathbf{\Sigma}^{-1}|$$

In [22], the $-\log|\mathbf{\Sigma}^{-1}|$ term is dropped since the authors solve their cost function in terms of $\mathbf{\Sigma}$ and this term is concave in $\mathbf{\Sigma}$. A constraint of $\mathrm{tr}(\mathbf{\Sigma}) = 1$ was introduced in its place to restrict the complexity of the model. We keep the $\log|\cdot|$ constraint in our cost function since we parameterize our model in terms of $\mathbf{\Sigma}^{-1}$, and $-\log|\mathbf{\Sigma}^{-1}|$ is convex in $\mathbf{\Sigma}^{-1}$.

### 3.2 Optimization

We take an alternating optimization approach to solve the optimization problem given by Equation 6. Each sub-problem in the alternating optimization steps is convex. The matrices $\mathbf{\Sigma}$ and $\mathbf{\Omega}$ are initialized to $\mathbf{I}$ in the beginning. The bias vector $\mathbf{b}$ is initialized to $\frac{1}{N}\mathbf{Y}^{\top}\mathbf{1}$.

**Optimization w.r.t. W when $\mathbf{\Omega}^{-1}, \mathbf{\Sigma}^{-1}$ and b are fixed:**

Given $\mathbf{\Omega}^{-1}, \mathbf{\Sigma}^{-1}, \mathbf{b}$, the matrix $\mathbf{W}$ consisting of the regression weight vectors of all the tasks can be obtained by solving the following optimization problem:

$$\underset{\mathbf{W}}{\arg\min}\ \mathrm{tr}((\mathbf{Y} - \mathbf{XW} - \mathbf{1b}^{\top})\mathbf{\Omega}^{-1}(\mathbf{Y} - \mathbf{XW} - \mathbf{1b}^{\top})^{\top}) + \lambda\,\mathrm{tr}(\mathbf{WW}^{\top}) + \lambda_1\,\mathrm{tr}(\mathbf{W\Sigma}^{-1}\mathbf{W}^{\top}) \tag{7}$$

The estimate $\hat{\mathbf{W}}$ is given by solving the following system of linear equations w.r.t. $\mathbf{W}$:

$$\left[(\boldsymbol{\Omega}^{-1} \otimes \mathbf{X}'\mathbf{X}) + \left((\lambda_1 \boldsymbol{\Sigma}^{-1} + \lambda \mathbf{I}_K) \otimes \mathbf{I}_D\right)\right] \text{vec}(\mathbf{W}) = \text{vec}(\mathbf{X}'(\mathbf{Y} - \mathbf{1}\mathbf{b}^\top)\boldsymbol{\Omega}^{-1}) \tag{8}$$

It is easy to see that with $\boldsymbol{\Omega}$ and $\boldsymbol{\Sigma}$ set to identity, the model becomes equivalent to solving $K$ regularized independent linear regression problems.

**Optimization w.r.t. b when $\boldsymbol{\Omega}^{-1}, \boldsymbol{\Sigma}^{-1}$ and W are fixed:**

Given $\boldsymbol{\Omega}^{-1}, \boldsymbol{\Sigma}^{-1}, \mathbf{W}$, the bias vector $\mathbf{b}$ for all the $K$ outputs can be obtained by solving the following optimization problem:

$$\arg\min_{\mathbf{b}} \text{tr}((\mathbf{Y} - \mathbf{X}\mathbf{W} - \mathbf{1}\mathbf{b}^\top)\boldsymbol{\Omega}^{-1}(\mathbf{Y} - \mathbf{X}\mathbf{W} - \mathbf{1}\mathbf{b}^\top)^\top) \tag{9}$$

The estimate $\hat{\mathbf{b}}$ is given by $\hat{\mathbf{b}} = \frac{1}{N}\sum_{n=1}^{N}(\mathbf{Y} - \mathbf{X}\mathbf{W})^\top \mathbf{1}$

**Optimization w.r.t. $\boldsymbol{\Sigma}^{-1}$ when $\boldsymbol{\Omega}^{-1}, \mathbf{W}$ and b are fixed:**

Given $\boldsymbol{\Omega}^{-1}, \mathbf{W}, \mathbf{b}$, the task inverse covariance matrix $\boldsymbol{\Sigma}^{-1}$ can be estimated by solving the following optimization problem:

$$\arg\min_{\boldsymbol{\Sigma}^{-1}} \lambda_1 \text{tr}(\mathbf{W}\boldsymbol{\Sigma}^{-1}\mathbf{W}^\top) - D\log|\boldsymbol{\Sigma}^{-1}| + \lambda_3||\boldsymbol{\Sigma}^{-1}||_1 \tag{10}$$

It is easy to see that the above is an instance of the standard inverse covariance estimation problem with sample covariance $\frac{\lambda_1}{D}\mathbf{W}^\top\mathbf{W}$, and can be solve using standard tools for inverse covariance estimation. We use the graphical Lasso procedure [8] to solve Equation 10 to estimate $\boldsymbol{\Sigma}^{-1}$:

$$\hat{\boldsymbol{\Sigma}}^{-1} = \text{gLasso}(\frac{\lambda_1}{D}\mathbf{W}^\top\mathbf{W}, \lambda_3) \tag{11}$$

If we assume $\boldsymbol{\Sigma}^{-1}$ to be *non-sparse*, we can drop the $\ell_1$ penalty on $\boldsymbol{\Sigma}^{-1}$ from Equation 10. However, the solution to $\boldsymbol{\Sigma}^{-1}$ will not be defined (when $K > D$) or will overfit (when $K$ is of the same order as $D$). To avoid this, we add a regularizer of the form $\lambda \text{tr}(\boldsymbol{\Sigma}^{-1})$ to Equation 10. This can be seen as imposing a matrix variate Gaussian prior on $\boldsymbol{\Sigma}^{-1/2}$ with both row and column covariance matrices equal to $\mathbf{I}$ to make the solution well defined. In the previous case of sparse $\boldsymbol{\Sigma}^{-1}$, the solution was well defined because of the sparsity prior on $\boldsymbol{\Sigma}^{-1}$. The optimization problem for $\boldsymbol{\Sigma}^{-1}$ is then given as

$$\arg\min_{\boldsymbol{\Sigma}^{-1}} \lambda_1 \text{tr}(\mathbf{W}\boldsymbol{\Sigma}^{-1}\mathbf{W}^\top) - D\log|\boldsymbol{\Sigma}^{-1}| + \lambda \text{tr}\left(\boldsymbol{\Sigma}^{-1}\right). \tag{12}$$

Equation 12 admits a closed form solution which is given by $\left(\frac{\lambda_1 \mathbf{W}^\top\mathbf{W} + \lambda\mathbf{I}}{D}\right)^{-1}$. For the non-sparse $\boldsymbol{\Sigma}^{-1}$ case, we keep the parameter $\lambda$ same as the hyperparameter for the term $\text{tr}(\mathbf{W}\mathbf{W}^\top)$ in Equation 6.

**Optimization w.r.t. $\boldsymbol{\Omega}^{-1}$ when $\boldsymbol{\Sigma}^{-1}, \mathbf{W}$ and b are fixed:**

Given $\boldsymbol{\Sigma}^{-1}, \mathbf{W}, \mathbf{b}$, the task inverse covariance matrix $\boldsymbol{\Omega}^{-1}$ can be estimated by solving the following optimization problem:

$$\arg\min_{\boldsymbol{\Omega}^{-1}} \text{tr}((\mathbf{Y} - \mathbf{X}\mathbf{W} - \mathbf{1}\mathbf{b}^\top)\boldsymbol{\Omega}^{-1}(\mathbf{Y} - \mathbf{X}\mathbf{W} - \mathbf{1}\mathbf{b}^\top)^\top) - N\log|\boldsymbol{\Omega}^{-1}| + \lambda_2||\boldsymbol{\Omega}^{-1}||_1 \tag{13}$$

It is again easy to see that the above problem is an instance of the standard inverse covariance estimation problem with sample covariance $\frac{1}{N}(\mathbf{Y} - \mathbf{X}\mathbf{W} - \mathbf{1}\mathbf{b}^\top)'(\mathbf{Y} - \mathbf{X}\mathbf{W} - \mathbf{1}\mathbf{b}^\top)$, and can be solved using standard tools for inverse covariance estimation. We use the graphical Lasso procedure [8] to solve Equation 10 to estimate $\boldsymbol{\Sigma}^{-1}$:

$$\hat{\boldsymbol{\Omega}}^{-1} = \text{gLasso}(\frac{1}{N}(\mathbf{Y} - \mathbf{X}\mathbf{W} - \mathbf{c}\mathbf{b}^\top)^\top(\mathbf{Y} - \mathbf{X}\mathbf{W} - \mathbf{c}\mathbf{b}^\top), \lambda_2) \tag{14}$$

## 4   Experiments

In this section, we evaluate our model by comparing it with several relevant baselines on both synthetic and real-world datasets. Our main set of results are on multiple-output regression problems on which we report mean-squared errors averaged across all the outputs. However, since our model also provides an estimate of the *conditional* inverse covariance structure $\boldsymbol{\Omega}^{-1}$ of the outputs, in Section 4.3 we provide experimental results on the structure recovery task as well. We compare our method with following baselines:

- **Independent regressions (RLS):** This baseline learns regularized least squares (RLS) regression model for each output, without assuming any structure among the weight vectors or among the outputs. This corresponds to our model with $\mathbf{\Sigma} = \mathbf{I}_K$ and $\mathbf{\Omega} = \mathbf{I}_K$. The weight vector of each individual problem is $\ell_2$ regularized with a hyperparameter $\lambda$.
- **Curds and Whey (C&W):** The predictor in Curds and Whey [3] takes the form $\mathbf{W}_{cw} = \mathbf{W}_{rls}\mathbf{U}\mathbf{\Lambda}\mathbf{U}^-$, where $\mathbf{W}_{rls}$ denotes the regularized least squares predictor, the columns of matrix $\mathbf{U}$ are the projection directions for the responses $Y$ obtained from canonical correlation analysis (CCA) of $\mathbf{X}$ and $\mathbf{Y}$, and $\mathbf{U}^-$ denotes Moore-Penrose pseudoinverse of $\mathbf{U}$. The diagonal matrix $\mathbf{\Lambda}$ contains the shrinkage factors for each CCA projection direction.
- **Multi-task Relationship Learning (MTRL):** This method leverages task relationships by assuming a matrix-variate prior on the weight matrix $\mathbf{W}$ [22]. We chose this baseline because of its flexibility in modeling the task relationships by "discovering" how the weight vectors are related (via $\mathbf{\Sigma}^{-1}$), rather than assuming a specific structure on them such as shared sparsity [16], low-rank assumption [2], etc. However MTRL in the multiple-output regression setting cannot take into account the output structure. It is therefor a special case of our model if we assume the output inverse covariance matrix $\mathbf{\Omega}^{-1} = \mathbf{I}$. The MTRL approach proposed in [22] does not have sparse penalty on $\mathbf{\Sigma}^{-1}$. We experimented with both sparse and non-sparse variants of MTRL and report the better of the two results here.
- **Multivariate Regression with Covariance Estimation (MRCE-$\ell_2$):** This baseline is the $\ell_2$ regularized variant of the MRCE model [17]. MRCE leverages output structure by assuming a full noise covariance in multiple-output regression and learning it along with the weight matrix $\mathbf{W}$ from the data. MRCE however cannot take into account the task structure because it cannot capture the relationships among the columns of $\mathbf{W}$. It is therefore a special case of our model if we assume the task inverse covariance matrix $\mathbf{\Sigma}^{-1} = \mathbf{I}$. We do not compare with the original $\ell_1$ regularized MRCE [17] to ensure a fair comparison by keeping all the models non-sparse in weight vectors.

In the experiments, we refer to our model as **MROTS** (**M**ultiple-output **R**egression with **O**utput and **T**ask **S**tructures). We experiment with two variants of our proposed model, one *without* a sparsity inducing penalty on the task coupling matrix $\mathbf{\Sigma}^{-1}$ (called **MROTS-I**), and the other *with* the sparse penalty on $\mathbf{\Sigma}^{-1}$ (called **MROTS-II**). The hyperparameters are selected using four-fold cross-validation. Both MTRL and MRCE-$\ell_2$ have two hyperparameters each and these are selected by searching on a two-dimensional grid. For the proposed model with non-sparse $\mathbf{\Sigma}^{-1}$, we fix the hyperparameter $\lambda$ in Equations 6 and 12 as $0.001$ for all the experiments. This is used to ensure that the task inverse covariance matrix estimate $\hat{\mathbf{\Sigma}}^{-1}$ exists and is robust when number of response variables $K$ is of the same order or larger than the input dimension $D$. The other two parameters $\lambda_1$ and $\lambda_2$ are selected using cross-validation. For sparse $\mathbf{\Sigma}^{-1}$ case, we use the same values of $\lambda_1$ and $\lambda_2$ that were selected for non-sparse case, and only the third parameter $\lambda_3$ is selected by cross-validation. This procedure avoids a potentially expensive search over a three dimensional grid. The hyperparameter $\lambda$ in Equation 6 is again fixed at $0.001$.

## 4.1 Synthetic data

We describe the process for synthetic data generation here. First, we generate a random positive definite matrix $\mathbf{\Sigma}^{-1}$ which will act as the task inverse covariance matrix. Next, a matrix $\mathbf{V}$ of size $D \times K$ is generated with each entry sampled from a zero mean and $1/D$ variance normal distribution. We compute the square-root $\mathbf{S}$ of $\mathbf{\Sigma}$ ($= \mathbf{S}\mathbf{S}$, where $\mathbf{S}$ is also a symmetric positive definite matrix), and $\mathbf{S}$ is used to generate the final weight matrix $\mathbf{W}$ as $\mathbf{W} = \mathbf{V}\mathbf{S}$. It is clear that for a $\mathbf{W}$ generated in this fashion, we will have $E[\mathbf{W}^T\mathbf{W}] = \mathbf{S}\mathbf{S} = \mathbf{\Sigma}$. This process generates $\mathbf{W}$ such that its columns (and therefore the weight vectors for different outputs) are correlated. A bias vector $\mathbf{b}$ of size $K$ is generated randomly from a zero mean unit variance normal distribution. Then we generate a sparse random positive definite matrix $\mathbf{\Omega}^{-1}$ that acts as the conditional inverse covariance matrix on output noise making the outputs correlated (given the inputs). Next, input samples are generated i.i.d. from a normal distribution and the corresponding multivariate output variables are generated as $\mathbf{y}_i = \mathbf{W}\mathbf{x}_i + \mathbf{b} + \boldsymbol{\epsilon}_i$, $\forall i = 1, 2, \ldots, N$, where $\boldsymbol{\epsilon}_i$ is the correlated noise vector randomly sampled from a zero mean normal distribution with covariance matrix $\mathbf{\Omega}$.

We generate three sets of synthetic data using the above process to gauge the effectiveness of the proposed model under varying circumstances: (i) $D = 20$, $K = 10$ and non-sparse $\mathbf{\Sigma}^{-1}$, (ii)

| Method | Synth data I | Synth data II | Synth data III | Paper I | Paper II | Gene data |
|--------|-------------|---------------|----------------|---------|----------|-----------|
| RLS | 37.29 | 3.22 | 3.94 | 1.08 | 1.04 | 1.92 |
| C&W | 37.14 | 21.88 | 7.06 | 1.08 | 1.08 | 1.51 |
| MTRL | 34.45 | 3.12 | 3.86 | 1.07 | 1.03 | 1.24 |
| MRCE-$\ell_2$ | 29.84 | 3.08 | 3.92 | 1.36 | 1.03 | 1.55 |
| MROTS-I | 26.65 | 2.61 | 3.75 | **0.90** | 1.03 | **1.18** |
| MROTS-II | **25.90** | **2.60** | **3.55** | **0.90** | 1.03 | 1.20 |

Table 1: Prediction error (MSE) on synthetic and real datasets. RLS: Independent regression, C&W: Curds and Whey model [3], MTRL: Multi-task relationship learning [22], MRCE-$\ell_2$: The $\ell_2$-regularized version of MRCE [17], MROTS-I: our model without sparse penalty on $\mathbf{\Sigma}^{-1}$, MROTS-II: our model with sparse penalty on $\mathbf{\Sigma}^{-1}$. Best results are highlighted in bold fonts.

$D = 10$, $K = 20$ and non-sparse $\mathbf{\Sigma}^{-1}$, and (iii) $D = 10$, $K = 20$ and sparse $\mathbf{\Sigma}^{-1}$. We also experiment with varying number of training samples ($N = 20, 30, 40$ and $50$).

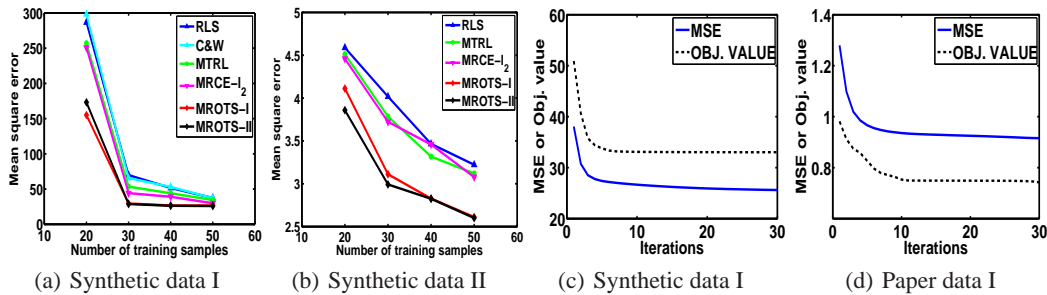

(a) Synthetic data I     (b) Synthetic data II     (c) Synthetic data I     (d) Paper data I

Figure 1: (a) and (b): Mean Square Error with varying number of training samples, (c) and (d): Mean Square Error and the value of the Objective function with increasing iterations for the proposed method.

## 4.2 Real data

We also evaluate our model on the following real-world multiple-output regression datasets:

- **Paper datasets:** These are two multivariate multiple-response regression datasets from paper industry [1]. The first dataset has 30 samples with each sample having 9 features and 32 outputs. The second dataset has 29 samples (after ignoring one sample with missing response variables), each having 9 features and 13 outputs. We take 15 samples for training and the remaining samples for test.
- **Genotype dataset:** This dataset has genotypes as input variables and phenotypes or observed traits as output variables [12]. The number of genotypes (features) is 25 and the number of phenotypes (outputs) is 30. We have a total of 100 samples in this dataset and we split it equally into training and test data.

The results on synthetic and real-world datasets are shown in Table 1. For synthetic datasets, the reported results are with 50 training samples. Independent linear regression performs the worst on all synthetic datasets. MRCE-$\ell_2$ performs better than MTRL on first and second synthetic data while MTRL is better on the third dataset. This mixed behavior of MRCE-$\ell_2$ and MTRL supports our motivation that both task structure (i.e., relationships among weight vectors) and output structure are important in multiple-output regression. Both MTRL and MRCE-$\ell_2$ are special cases of our model with former ignoring the output structure (captured by $\mathbf{\Omega}^{-1}$) and the latter ignoring the weight vector relationships (captured by $\mathbf{\Sigma}^{-1}$). Both variants of our model (MROTS-I and MROTS-II) perform significantly better than the compared baselines. The improvement with sparse $\mathbf{\Sigma}^{-1}$ variant is more prominent on the third dataset which is generated with sparse $\mathbf{\Sigma}^{-1}$ (5.33% relative reduction in MSE), than on the first two datasets (2.81% and 0.3% relative reduction in MSE). However, in our experiments, the sparse $\mathbf{\Sigma}^{-1}$ variant (MROTS-II) always performed better or as good as the non-sparse variant on all synthetic and real datasets, which suggests that *explicitly* encouraging zero entries in $\mathbf{\Sigma}^{-1}$ leads to better estimates of task relationships (by avoiding spurious correlations between weight vectors). This can potentially improve the prediction performance. Finally, we also note that the Curds & Whey method [3] performs significantly worse than RLS for Synthetic data II and III. C&W uses CCA to project the response matrix $\mathbf{Y}$ to a lower $\min(D, K)$-dimensional space learning $\min(D, K)$ predictors there and then projecting them back to the original K-dimensional

space. This procedure may end up throwing away relevant information from responses if $K$ is much higher than $D$. These empirical results suggest that C&W may adversely affect the prediction performance when the number of response variables $K$ is higher than the number of explanatory variables $D$ ($D = 2K$ in these cases).

On the real-world datasets too, our model performs better than or on par with the compared baselines. Both MROTS-I and MROTS-II perform significantly better than the other baselines on the first Paper dataset (9 features and 32 outputs per sample). All models perform almost similarly on the second Paper dataset (9 features and 13 outputs per sample), which could be due to the absence of a strong task or output structure in this data. C&W does not preform well on both Paper datasets which might be due to the reason discussed earlier. On the genotype-phenotype prediction task too, both our models achieve better average mean squared errors than the other baselines, with both variants performing roughly comparably.

We also evaluate our model's performance with varying number of training examples and compare with the other baselines. Figures 1(a) and 1(b) show the plots of mean square error vs. number of training examples for first two synthetic datasets. We do not plot C&W for Synthetic data II since it performs worse than RLS. On the first synthetic data, the performance gain of our model is more pronounced when number of training examples is small. For the second synthetic data, we retain similar performance gain over other models when number of training examples are increased from 20. The MSE numbers for the first synthetic data are higher than the ones obtained for the second synthetic data because of a difference in the magnitude of error covariances used in the generation of datasets.

We also experiment with the convergence properties of our method. Figures 1(c) and 1(d) show that plots of average MSE and the value of the objective function (given by Equation 6) with increasing number of iterations on the first synthetic dataset and the first Paper dataset. The plots show that our alternating optimization procedure converges in roughly 10–15 iterations.

### 4.3 Covariance structure recovery

Although not the main goal of the paper, we experiment with learned inverse covariance matrix of the outputs (given the inputs) as a sanity check on the proposed model. To better visualize, we generate a dataset with 5 responses and 3 predictors using the same process as described in Sec. 4.1. Figure on the right shows the true conditional inverse covariance matrix $\Omega^{-1}$ (Top), the matrix learned with MROTS $\hat{\Omega}^{-1}$ (Middle), and the precision matrix learned with graphical lasso ignoring the predictors (Bottom). Taking into account the regression weights results in better estimate of the true covariance matrix. We got similar results for MRCE-$\ell_2$ which also takes into account the predictors while learning the inverse covariance, although MROTS estimates were closer to the ground truth in terms of the Frobenius norm.

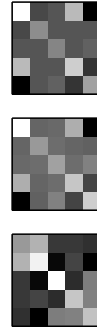

## 5 Related Work

Apart from the prior works discussed in Section 1, our work has connections to some other works which we discuss in this section. Recently, Sohn & Kim [18] proposed a model for jointly estimating the weight vector for each output and the covariance structure of the outputs. However, they assume a shared sparsity structure on the weight vectors. This assumption may be restrictive in some problems. Some other works on conditional graphical model estimation [20, 4] are based on similar structural sparsity assumptions. In contrast, our model does not assume any specific structure on the weight vectors, and by explicitly modeling the covariance structure of the weight vectors, *learns* the appropriate underlying structure from the data.

## 6 Future Work and Conclusion

We have presented a flexible model for multiple-output regression taking into account the covariance structure of the outputs and the covariance structure of the underlying prediction tasks. Our model does not require a priori knowledge of these structures and learns these from the data. Our model leads to improved accuracies on multiple-output regression tasks. Our model can be extended in several ways. For example, one possibility is to model nonlinear input-output relationships by kernelizing the model along the lines of [22].

# References

[1] M. Aldrin. Moderate projection pursuit regression for multivariate response data. *Computational Statistics and Data Analysis*, 21, 1996.

[2] Andreas Argyriou, Theodoros Evgeniou, and Massimiliano Pontil. Multi-task feature learning. In *NIPS*, 2007.

[3] L. Breiman and J.H. Friedman. Predicting multivariate responses in multiple linear regression. *Journal of the Royal Statistical Society. Series B (Methodological)*, pages 3–54, 1997.

[4] T. Cai, H. Li, W. Liu, and J. Xie. Covariate adjusted precision matrix estimation with an application in genetical genomics. *Biometrika*, 2011.

[5] Rich Caruana. Multitask Learning. *Machine Learning*, 28, 1997.

[6] J. Cheng, E. Levina, P. Wang, and J. Zhu. Sparse ising models with covariates. *arXiv:1209.6342v1*, 2012.

[7] S. Ding, G. Wahba, and J. X. Zhu. Learning higher-order graph structure with features by structure penalty. In *NIPS*, 2011.

[8] J. Friedman, T. Hastie, and R. Tibshirani. Sparse inverse covariance estimation with the graphical lasso. *Biostatistics*, 9(3):432–441, 2008.

[9] P. Goovaerts. *Geostatistics For Natural Resources Evaluation*. Oxford University Press, 1997.

[10] T. Heskes. Empirical Bayes for learning to learn. *ICML*, 2000.

[11] S. Kim, K. Sohn, and E. P. Xing. A multivariate regression approach to association analysis of a quantitative trait network.

[12] S. Kim and E. P. Xing. Statistical estimation of correlated genome associations to a quantitative trait network. *PLoS Genetics*, 2009.

[13] S. Kim and E. P. Xing. Tree-guided group lasso for multi-response regression with structured sparsity, with an application to eQTL mapping. *Annals of Applied Statistics*, 2012.

[14] W. Lee and Y. Liu. Simultaneous multiple response regression and inverse covariance matrix estimation via penalized gaussian maximum likelihood. *Journal of Multivariate Analysis*, 2012.

[15] H. Liu, X. Chen, J. Lafferty, and L. Wasserman. Graph-valued regression. In *NIPS*, 2010.

[16] G. Obozinskiy, M. J. Wainwright, and M. I. Jordan. Union support recovery in high-dimensional multivariate regression. In *NIPS*, 2010.

[17] A. J. Rothman, E. Levina, and J. Zhu. Sparse multivariate regression with covariance estimation. *Journal of Computational and Graphical Statistics*, 2010.

[18] K.A. Sohn and S. Kim. Joint estimation of structured sparsity and output structure in multiple-output regression via inverse-covariance regularization. In *AISTATS*, 2012.

[19] R. Tibshirani. Regression shrinkage and selection via the lasso. *Journal of Royal Statistical Society*, 1996.

[20] J. Yin and H. Li. A sparse conditional gaussian graphical model for analysis of genetical genomics data. *The Annals of Applied Statistics*, 2011.

[21] Y. Zhang and J. Schneider. Learning Multiple Tasks with a Sparse Matrix-Normal Penalty. In *NIPS*, 2010.

[22] Y. Zhang and D. Yeung. A convex formulation for learning task relationships in multi-task learning. In *UAI*, 2010.

[23] S. Zhou, J. Lafferty, and L. Wasserman. Time varying undirected graphs. *Machine Learning Journal*, 2010.

